# On a Modification to the Mean Field EM Algorithm in Factorial Learning

**A. P. Dunmur**      **D. M. Titterington**
Department of Statistics
Maths Building
University of Glasgow
Glasgow G12 8QQ, UK

alan@stats.gla.ac.uk      mike@stats.gla.ac.uk

## Abstract

A modification is described to the use of mean field approximations in the E step of EM algorithms for analysing data from latent structure models, as described by Ghahramani (1995), among others. The modification involves second-order Taylor approximations to expectations computed in the E step. The potential benefits of the method are illustrated using very simple latent profile models.

## 1 Introduction

Ghahramani (1995) advocated the use of mean field methods as a means to avoid the heavy computation involved in the E step of the EM algorithm used for estimating parameters within a certain latent structure model, and Ghahramani & Jordan (1995) used the same ideas in a more complex situation. Dunmur & Titterington (1996a) identified Ghahramani's model as a so-called latent profile model, they observed that Zhang (1992, 1993) had used mean field methods for a similar purpose, and they showed, in a simulation study based on very simple examples, that the mean field version of the EM algorithm often performed very respectably. By this it is meant that, when data were generated from the model under analysis, the estimators of the underlying parameters were efficient, judging by empirical results, especially in comparison with estimators obtained by employing the 'correct' EM algorithm: the examples therefore had to be simple enough that the correct EM algorithm is numerically feasible, although any success reported for the mean field

version is, one hopes, an indication that the method will also be adequate in more complex situations in which the correct EM algorithm is not implementable because of computational complexity.

In spite of the above positive remarks, there were circumstances in which there was a perceptible, if not dramatic, lack of efficiency in the simple (naive) mean field estimators, and the objective of this contribution is to propose and investigate ways of refining the method so as to improve performance without detracting from the appealing, and frequently essential, simplicity of the approach. The procedure used here is based on a second order correction to the naive mean field well known in statistical physics and sometimes called the cavity or TAP method (Mezard, Parisi & Virasoro , 1987). It has been applied recently in cluster analysis (Hofmann & Buhmann, 1996). In Section 2 we introduce the structure of our model, Section 3 explains the refined mean field approach, Section 4 provides numerical results, and Section 5 contains a statement of our conclusions.

## 2   The Model

The model under study is a latent profile model (Henry, 1983), which is a latent structure model involving continuous observables $\{x_r : r = 1 \ldots p\}$ and discrete latent variables $\{y_i : i = 1 \ldots d\}$. The $y_i$ are represented by indicator vectors such that for each $i$ there is a single $j$ such that $y_{ij} = 1$ and $y_{ik} = 0$, for all $k \neq j$. The latent variables are connected to the observables by a set of weight matrices $W_i$ in such a way that the distribution of the observations given the latent variables is a multivariate Gaussian with mean $\sum_i W_i y_i$ and covariance matrix $\Gamma$. To ease the notation, the covariance matrix is taken to be the identity matrix, although extension is quite easy to the case where $\Gamma$ is a diagonal matrix whose elements have to be estimated (Dunmur & Titterington, 1996a). Also to simplify the notation, the marginal distributions of the latent variables are taken to be uniform, so that the totality of unknown parameters is made up of the set of weight matrices, to be denoted by $W = (W_1, W_2, \ldots, W_d)$.

## 3   Methodology

In order to learn about the model we have available a dataset $\mathcal{D} = \{x^\mu : \mu = 1 \ldots N\}$ of $N$ independent, $p$ dimensional realizations of the model, and we adopt the Maximum Likelihood approach to the estimation of the weight matrices. As is typical of latent structure models, there is no explicit Maximum Likelihood estimate of the parameters of the model, but there is a version of the EM algorithm (Dempster, Laird & Rubin, 1977) that can be used to obtain the estimates numerically. The EM algorithm consists of a sequence of double steps, E steps and M steps.

At stage $m$ the E step, based on a current estimate $W^{m-1}$ of the parameters, calculates

$$\mathcal{Q}(W, W^{m-1}) = \langle \mathcal{L}_c(W) | \mathcal{D}, W^{m-1} \rangle ,$$

where, the expectation $\langle \cdot \rangle$ is over the latent variables $y$, and is conditional on $\mathcal{D}$ and $W^{m-1}$, and $\mathcal{L}_c$ denotes the crucial part of the *complete-data* log-likelihood, given

by

$$\mathcal{L}_c(W) = -\frac{1}{2}\sum_\mu \left(x^\mu - \sum_i W_i y_i^\mu\right)^T \left(x^\mu - \sum_j W_j y_j^\mu\right).$$

The M step then maximizes $Q$ with respect to $W$ and gives the new parameter estimate $W^m$.

For the simple model considered here, the M step gives

$$W^m = \left(\sum_\mu x^\mu \left\langle Y^{\mu T}\right\rangle\right)\left(\sum_\mu \left\langle Y^\mu Y^{\mu T}\right\rangle\right)^{-1}$$

where $W = (W_1, W_2, \ldots, W_d)$ and $Y^T = (y_1^T, y_2^T, \ldots, y_d^T)$ and, for brevity, explicit mention of the conditioned quantities in the expectations $\langle \cdot \rangle$ has been omitted. The above formula differs somewhat from that given by Ghahramani (1995).

Hence we need to evaluate the sets of expectations $\langle y_i \rangle$ and $\langle y_i y_j^T \rangle$ for each example in the dataset. (The superscript $\mu$ is omitted, for clarity.) As pointed out in Ghahramani (1995), it is possible to evaluate these expectations directly by summing over all possible latent states. This has the disadvantage of becoming exponentially more expensive as the size of the latent space increases.

The mean field approximation is well known in physics and can be used to reduce the computational complexity. At its simplest level, the mean field approximation replaces the joint expectations of the latent variables by the products of the individual expectations; this can be interpreted as bounding the likelihood from below (Saul, Jaakkola, Jordan, 1996). Here we consider a second order approximation, as outlined below.

Since the latent variables are categorical, it is simple to sum over the state space of a single latent variable. Hence, following Parisi (1988), the expectations of the latent variables are given by

$$\langle y_{ij} \rangle = \langle f_j(\epsilon_i) \rangle, \tag{1}$$

where $f_j(\epsilon_i)$ is the $j^{\text{th}}$ component of the softmax function; $\exp(\epsilon_{ij})/\sum_k \exp(\epsilon_{ik})$, and the expectation $\langle \cdot \rangle$ is taken over the remaining latent variables. The vector $\epsilon_i = \{\epsilon_{ij}\}$ contains the log probabilities (up to a constant) associated with each category of the latent variable for each example in the data set. For the simple model under study $\epsilon_{ij}$ is given by

$$\epsilon_{ij} = \left\{W_i^T(x - \sum_{k\neq i} W_k y_k)\right\}_j - \frac{1}{2}(W_i^T W_i)_{jj}. \tag{2}$$

The expectation in (1) can be expanded in a Taylor series about the average, $\langle \epsilon_i \rangle$, giving

$$\langle y_{ij} \rangle = f_j(\langle \epsilon_i \rangle) + \frac{1}{2}\sum_{kl}\langle \Delta\epsilon_{ik}\Delta\epsilon_{il}\rangle \frac{\partial^2 f_j(\langle \epsilon_i \rangle)}{\partial \epsilon_{ik}\partial \epsilon_{il}} + \mathcal{O}(\Delta\epsilon^3), \tag{3}$$

where $\Delta\epsilon_{ij} = \epsilon_{ij} - \langle \epsilon_{ij} \rangle$. The *naive* mean field approximation simply ignores all corrections. We can postulate that the second order fluctuations are taken care of

by a so called *cavity field,* (see, for instance, Mezard, Parisi & Virasoro , 1987, p.16), that is,

$$\langle y_{ij} \rangle = f_j(\langle \epsilon_i \rangle + h_i), \tag{4}$$

where the vector of fields $h_i = \{h_{ik}\}$ has been introduced to take care of the correction terms. This equation may also be expanded in a Taylor series to give

$$\langle y_{ij} \rangle = f_j(\langle \epsilon_i \rangle) + \sum_k h_{ik} \frac{\partial f_j(\langle \epsilon_i \rangle)}{\partial \epsilon_{ik}} + \mathcal{O}(h^2).$$

Then, equating coefficients with (3) and after a little algebra, we get

$$h_{ij} = \frac{1}{2} \sum_k \langle \Delta\epsilon_{ij} \Delta\epsilon_{ik} \rangle (\delta_{jk} - 2f_k(\langle \epsilon_i \rangle)), \tag{5}$$

where $\delta_{jk}$ is the Kronecker delta and, for the model under consideration,

$$\langle \Delta\epsilon_{ij} \Delta\epsilon_{ik} \rangle = \left( W_i^T \sum_{mn \neq i} W_m (\langle y_m y_n^T \rangle - \langle y_m \rangle \langle y_n^T \rangle) W_n^T W_i \right)_{jk}. \tag{6}$$

The naive mean field assumption may be used in (6), giving

$$(\langle y_i y_j^T \rangle - \langle y_i \rangle \langle y_j^T \rangle)_{kl} = \delta_{ij} \langle y_{ik} \rangle (\delta_{kl} - \langle y_{il} \rangle). \tag{7}$$

Within the E step, for each realization in the data set, the mean fields (4), along with the cavity fields (5), can be evaluated by an iterative procedure which gives the individual expectations of the latent variables. The naive mean field approximation (7) is then used to evaluate the joint expectations $\langle y_i y_j^T \rangle$. In the next section we report, for a simple model, the effect on parameter estimation of the use of cavity fields.

## 4   Results

Simulations were carried out using latent variable models (i) with 5 observables and 4 binary hidden variables and (ii) with 5 observables and 3 3-state hidden variables. The weight matrices were generated from zero mean Gaussian variables with standard deviation $w$. In order to make the M step trivial it was assumed that the matrices were known up to the scale parameter $w$, and this is the parameter estimated by the algorithm. A data set was generated using the known parameter and this was then estimated using straight EM, naive mean field (MF) and mean field with cavity fields (MF$_{\text{cav}}$).

Although datasets of sizes 100 and 500 were generated, only the results for $N = 500$ are presented here since both scenarios showed the same qualitative behaviour. Also, the estimation algorithms were started from different initial positions; this too had no effect on the final estimates of the parameters. A representative selection of results follows; Table 1 shows the results for both the $5 \times 4 \times 2$ model and the $5 \times 3 \times 3$ model.

The results show that, when the true value, $w_{\text{tr}}$, of the parameter was small, there is little difference among the three methods. This is due to the fact that at these

Table 1: Table of results N=500, results averaged over 50 simulations for 5 observables with 4 binary latent variables and for 5 observables with 3 3-state latent variables. The figures in brackets give the standard deviation of the estimates, $w_{est}$, in units according to the final decimal place. RMS is the root mean squared error of the estimate compared to the true value.

| Method | $w_{tr}$ | $w_{init}$ | $5 \times 4 \times 2$ | | $5 \times 3 \times 3$ | |
|---|---|---|---|---|---|---|
| | | | $w_{est}$ | RMS | $w_{est}$ | RMS |
| EM | 0.1 | 0.05 | 0.09(1) | 0.014 | 0.10(2) | 0.024 |
| MF | 0.1 | 0.05 | 0.09(1) | 0.014 | 0.10(2) | 0.023 |
| MF$_{cav}$ | 0.1 | 0.05 | 0.09(1) | 0.014 | 0.10(2) | 0.023 |
| EM | 0.5 | 0.1 | 0.49(2) | 0.016 | 0.50(2) | 0.019 |
| MF | 0.5 | 0.1 | 0.47(2) | 0.029 | 0.46(2) | 0.038 |
| MF$_{cav}$ | 0.5 | 0.1 | 0.48(2) | 0.026 | 0.47(2) | 0.032 |
| EM | 1.0 | 0.1 | 0.99(2) | 0.016 | 1.00(2) | 0.018 |
| MF | 1.0 | 0.1 | 0.96(2) | 0.040 | 0.98(2) | 0.032 |
| MF$_{cav}$ | 1.0 | 0.1 | 0.99(2) | 0.018 | 1.00(2) | 0.018 |
| EM | 2.0 | 0.2 | 1.99(1) | 0.014 | 1.99(1) | 0.015 |
| MF | 2.0 | 0.2 | 1.98(2) | 0.021 | 1.98(2) | 0.023 |
| MF$_{cav}$ | 2.0 | 0.2 | 1.97(2) | 0.027 | 1.97(2) | 0.031 |
| EM | 5.0 | 0.1 | 4.99(1) | 0.013 | 5.00(1) | 0.013 |
| MF | 5.0 | 0.1 | 4.99(1) | 0.016 | 4.96(2) | 0.047 |
| MF$_{cav}$ | 5.0 | 0.1 | 4.97(2) | 0.032 | 4.88(3) | 0.114 |

small values there is little separation among the mixtures that are used to generate the data and hence the methods are all equally good (or bad) at estimating the parameter. As the true parameter increases and becomes close to one, the cavity field method performs significantly better then naive mean field; in fact it performs as well as EM for $w_{tr}=1$.

For values of $w_{tr}$ greater than one, the cavity field method performs less well than naive mean field. This suggests that the Taylor expansions (3) and (5) no longer provide reliable approximations. Since

$$\Delta\epsilon_{ij} = \left( \sum_{k\neq i} W_i^T W_k(\langle y_k \rangle - y_k) \right)_j , \qquad (8)$$

it is easy to see that if the elements in $W_i$ are much less than one then the corrections to the Taylor expansion are small and hence the cavity fields are small, so the approximations hold. If $w$ is much larger than one, then the mean field estimates become closer to zero and one since the energies $\epsilon_{ij}$ (equation 2) become more extreme. Hence if the mean fields correctly estimate the latent variables the corrections are indeed small, but if the mean fields incorrectly estimate the latent variable the error term is substantial, leading to a reduction in performance.

Another simulation, similar to that presented in Ghahramani (1995), was also studied to compare the modification to both the 'correct' EM and the naive mean field. The model has two latent variables that correspond to either horizontal or vertical

lines in one of four positions. These are combined and zero mean Gaussian noise added to produce a $4 \times 4$ example image. A data set is created from many of these examples with the latent variables chosen at random. From the data set the weight matrices connecting the observables to the latent variables are estimated and compared to the true weights which consist of zeros and ones. Typical results for a sample size of 160 and Gaussian noise of variance 0.2 are presented in Figure 1. The number of iterations needed to converge were similar for all three methods.

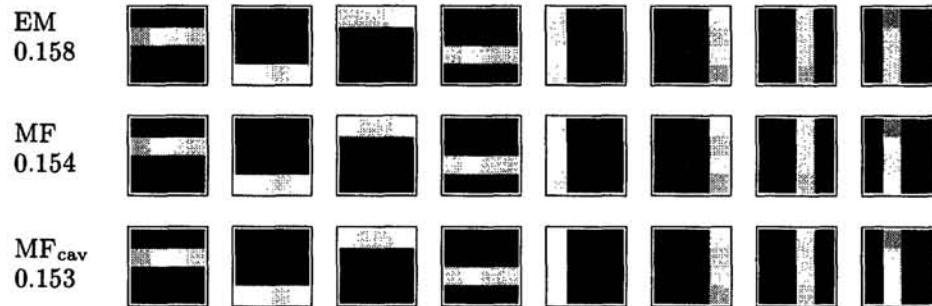

Figure 1: Estimated weights for a sample size of $N = 160$ and noise variance 0.2 added. The three rows correspond to EM, naive MF and $MF_{cav}$ respectively. The number on the left hand end of the row is the mean squared error of the estimated weights compared with the true weights. The first four images are the estimates of the first latent vector and the remaining four images are the estimates of second latent vector.

As can be seen from Figure 1 there is very little difference between the estimates of the weights and the mean squared errors are all very close. The mean field method converged in approximately four iterations which means that for this simple model the MF E step is taking approximately 32 steps as compared to 16 steps for the straight EM. This is due to the simplicity of the latent structure for this model. For a more complicated model the MF algorithm should take fewer iterations. Again the results are encouraging for the MF method, but they do not show any obvious benefit from using the cavity field correction terms.

## 5   Conclusion

The cavity field method can be applied successfully to improve the performance of naive mean field estimates. However, care must be taken when the corrections become large and actually degrade the performance. Predicting the failure modes of the algorithm may become harder for larger (more realistic) models. The message seems to be that where the mean field does well the cavity fields will improve the situation, but where the mean field performs less well the cavity fields can degrade performance. This suggests that the cavity fields could be used as a check on the mean field method. Where the cavity fields are small we can be reasonably confident that the mean field is producing sensible answers. However where the cavity fields become large it is likely that the mean field is no longer producing accurate estimates.

Further work would consider larger simulations using more realistic models. It

might no longer be feasible to compare these simulations with the 'correct' EM algorithm as the size of the model increases, though other techniques such as Gibbs sampling could be used instead. It would also be interesting to look at the next level of approximation where, instead of approximating the joint expectations by the product of the individual expectations in equation (6), the joint expectations are evaluated by summing over the joint state space (c.f. equation (1)) and possibly evaluating the corresponding cavity fields (Dunmur & Titterington, 1996b). This would perhaps improve the quality of the approximation without introducing the exponential complexity associated with the full E step.

## Acknowledgements

This research was supported by a grant from the UK Engineering and Physical Sciences Research Council.

# References

DEMPSTER, A. P., LAIRD, N. M. & RUBIN, D. B. (1977). Maximum likelihood estimation from incomplete data via the EM algorithm (with discussion). *J. R. Statist. Soc.* B **39**, 1-38.

DUNMUR, A. P. & TITTERINGTON, D. M. (1996a). Parameter estimation in latent structure models. *Tech. Report 96-2*, Dept. Statist., Univ. Glasgow.

DUNMUR, A. P. & TITTERINGTON, D. M. (1996b). Higher order mean field approximations. In preparation.

GHAHRAMANI, Z. (1995). Factorial learning and the EM algorithm. In *Advances in Neural Information Processing Systems 7*, Eds. G. Tesauro, D. S. Touretzky & T. K. Leen. Cambridge MA: MIT Press.

GHAHRAMANI, Z. & JORDAN, M. I. (1995). Factorial hidden Markov models. Computational Cognitive Science Technical Report 9502, MIT.

HENRY, N. W. (1983). Latent structure analysis. In *Encyclopedia of Statistical Sciences, Volume 4*, Eds. S. Kotz, N. L. Johnson & C. B. Read, pp.497-504. New York: Wiley.

HOFMANN, T. & BUHMANN, J. M. (1996) Pairwise Data Clustering by Deterministic Annealing. Tech. Rep. IAI-TR-95-7, Institut für Informatik III, Universität Bonn.

MEZARD, M., PARISI, G. & VIRASORO, M. A. (1987) *Spin Glass Theory and Beyond*. Lecture Notes in Physics, 9. Singapore: World Scientific.

PARISI, G. (1988). *Statistical Field Theory*. Redwood City CA: Addison-Wesley.

SAUL, L. K., JAAKKOLA, T. & JORDAN, M. I. (1996) Mean Field Theory for Sigmoid Belief Networks. *J. Artificial Intelligence Research* **4**, 61-76.

ZHANG, J. (1992). The Mean Field Theory in EM procedures for Markov random fields. *I. E. E. E. Trans. Signal Processing* **40**, 2570-83.

ZHANG, J. (1993). The Mean Field Theory in EM procedures for blind Markov random field image restoration. *I. E. E. E. Trans. Image Processing* **2**, 27-40.